# Variable KD-Tree Algorithms for Spatial Pattern Search and Discovery

**Jeremy Kubica**
Robotics Institute
Carnegie Mellon University
Pittsburgh, PA 15213
jkubica@ri.cmu.edu

**Joseph Masiero**
Institute for Astronomy
University of Hawaii
Honolulu, HI 96822
masiero@ifa.hawaii.edu

**Andrew Moore**
Robotics Institute
Carnegie Mellon University
Pittsburgh, PA 15213
awm@cs.cmu.edu

**Robert Jedicke**
Institute for Astronomy
University of Hawaii
Honolulu, HI 96822
jedicke@ifa.hawaii.edu

**Andrew Connolly**
Physics & Astronomy Department
University of Pittsburgh
Pittsburgh, PA 15213
ajc@phyast.pitt.edu

## Abstract

In this paper we consider the problem of finding sets of points that conform to a given underlying model from within a dense, noisy set of observations. This problem is motivated by the task of efficiently linking faint asteroid detections, but is applicable to a range of spatial queries. We survey current tree-based approaches, showing a trade-off exists between single tree and multiple tree algorithms. To this end, we present a new type of multiple tree algorithm that uses a variable number of trees to exploit the advantages of both approaches. We empirically show that this algorithm performs well using both simulated and astronomical data.

## 1 Introduction

Consider the problem of detecting faint asteroids from a series of images collected on a single night. Inherently, the problem is simply one of connect-the-dots. Over a single night we can treat the asteroid's motion as linear, so we want to find detections that, up to observational errors, lie along a line. However, as we consider very faint objects, several difficulties arise. First, objects near our brightness threshold may oscillate around this threshold, blinking into and out-of our images and providing only a small number of actual detections. Second, as we lower our detection threshold we will begin to pick up more spurious noise points. As we look for really dim objects, the number of noise points increases greatly and swamps the number of detections of real objects.

The above problem is one example of a model based spatial search. The goal is to identify sets of points that fit some given underlying model. This general task encompasses a wide range of real-world problems and spatial models. For example, we may want to detect a specific configuration of corner points in an image or search for multi-way structure in scientific data. We focus our discussion on problems that have a high density of both true

and noise points, but which may have only a few points actually from the model of interest. Returning to the asteroid linking example, this corresponds to finding a handful of points that lie along a line within a data set of millions of detections.

Below we survey several tree-based approaches for efficiently solving this problem. We show that both single tree and conventional multiple tree algorithms can be inefficient and that a trade-off exists between these approaches. To this end, we propose a new type of multiple tree algorithm that uses a *variable* number of tree nodes. We empirically show that this new algorithm performs well using both simulated and real-world data.

## 2 Problem Definition

Our problem consists of finding sets of points that fit a given underlying spatial model. In doing so, we are effectively looking for known types of structure buried within the data. In general, we are interested in finding sets with $k$ or more points, thus providing a sufficient amount of support to confirm the discovery. Finding this structure within the data may either be our end goal, such as in asteroid linkage, or may just be a preprocessor for a more sophisticated statistical test, such as renewal strings [1]. We are particularly interested in high-density, low-support domains where there may be many hundreds of thousands of points, but only a handful actually support our model.

Formally, the data consists of $N$ unique $D$-dimensional points. We assume that the underlying model can be estimated from $c$ unique points. Since $k \geq c$, the model may over-constrained. In these cases we divide the points into two sets: *Model Points* and *Support Points*. Model points are the $c$ points used to fully define the underlying model. Support points are the remaining points used to confirm the model. For example, if we are searching for sets of $k$ linear points, we could use a set's endpoints as model points and treat the middle $k-2$ as support points. Or we could allow any two points to serve as model points, providing an exhaustive variant of the RANSAC algorithm [2].

The prototypical example used in this paper is the (linear) asteroid linkage problem:

> For each pair of points find the $k-2$ best support points for the line that they define (such that we use at most one point at each time step).

In addition, we place restrictions on the validity of the initial pairs by providing velocity bounds. It is important to note that although we use this problem as a running example, the techniques described can be applied to a range of spatial problems.

## 3 Overview of Previous Approaches

### 3.1 Constructive Algorithms

Constructive algorithms "build up" valid sets of points by repeatedly finding additional points that are compatible with the current set. Perhaps the simplest approach is to perform a two-tiered *brute force* search. First, we exhaustively test all sets of $c$ points to determine if they define a valid model. Then, for each valid set we test all of the remaining points for support. For example in the asteroid linkage problem, we can initially search over all $O(N^2)$ pairs of points and for each of the resulting lines test all $O(N)$ points to determine if they support that line. A similar approach within the domain of target tracking is sequential tracking (for a good introduction see [3]), where points at early time steps are used to estimate a track that is then projected to later time steps to find additional support points.

In large-scale domains, these approaches can often be made tractable by using spatial structure in the data. Again returning to our asteroid example, we can place the points in a

KD-tree [4]. We can then limit the number of initial pairs examined by using this tree to find points compatible with our velocity constraints. Further, we can use the KD-tree to only search for support points in localized regions around the line, ignoring large numbers of obviously infeasible points. Similarly, trees have been used in tracking algorithms to efficiently find points near predicted track positions [5]. We call these adaptations *single tree* algorithms, because at any given time the algorithm is searching at most one tree.

### 3.2 Parameter Space Methods

Another approach is to search for valid sets of points by searching the model's parameter space, such as in the Hough transform [6]. The idea behind these approaches is that we can test whether each point is compatible with a small set of model parameters, allowing us to search parameter space to find the valid sets. However, this method can be expensive in terms of both computation and memory, especially for high dimensional parameter spaces. Further, if the model's total support is low, the true model occurrences may be effectively washed out by the noise. For these reasons we do not consider parameter space methods.

### 3.3 Multiple Tree Algorithms

The primary benefit of tree-based algorithms is that they are able to use spatial structure within the data to limit the cost of the search. However, there is a clear potential to push further and use structure from multiple aspects of the search *at the same time*. In doing so we can hopefully avoid many of the dead ends and wrong turns that may result from exploring bad initial associations in the first few points in our model. For example, in the domain of asteroid linkage we may be able to limit the number of short, initial associations that we have to consider by using information from later time steps. This idea forms the basis of multiple tree search algorithms [7, 8, 9].

Multiple tree methods explicitly search for the entire set of points at once by searching over *combinations* of tree nodes. In standard single tree algorithms, the search tries to find individual points satisfying some criteria (e.g. the next point to add) and the search state is represented by a single node that *could* contain such a point. In contrast, multiple tree algorithms represent the current search state with multiple tree nodes that *could* contain points that together conform to the model. Initially, the algorithm begins with $k$ root nodes from either the same or different tree data structures, representing the $k$ different points that must be found. At each step in the search, it narrows in on a set of mutually compatible spatial regions and thus a set of individual points that fit the model by picking one of the model nodes and recursively exploring its children. As with a standard "single tree" search, we constantly check for opportunities to prune the search.

There are several important drawbacks to multiple tree algorithms. First, additional trees introduce a higher branching factor in the search and increase the potential for taking deep "wrong turns." Second, care must be taken in order to deal with missing or a variable number of support points. Kubica *et. al.* discuss the use of an additional "missing" tree node to handle these cases [9]. However, this approach can effectively make repeated searches over subsets of trees, making it more expensive both in theory and practice.

## 4 Variable Tree Algorithms

In general we would like to exploit structural information from all aspects of our search problem, but do so while branching the search on just the parameters of interest. To this end we propose a new type of search that uses a *variable* number of tree nodes. Like a standard multiple tree algorithm, the variable tree algorithm searches combinations of tree nodes to find valid sets of points. However, we limit this search to just those points required

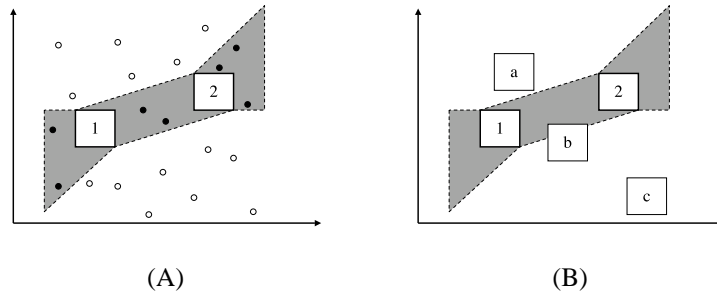

(A)                                        (B)

Figure 1: The model nodes' bounds (1 and 2) define a region of feasible support (shaded) for *any* combination of model points from those nodes (A). As shown in (B), we can classify entire support tree nodes as feasible (node b) or infeasible (nodes a and c).

to define, and thus bound, the models currently under consideration. Specifically, we use $M$ model tree nodes,[1] which guide the recursion and thus the search. In addition, throughout the search we maintain information about other potential *supporting* points that can be used to confirm the final track or prune the search due to a lack of support.

For example in the asteroid linking problem each line is defined by only 2 points, thus we can efficiently search through the models using a multiple tree search with 2 *model trees*. As shown in Figure 1.A, the spatial bounds of our current model nodes immediately limit the set of feasible support points for *all* line segments compatible with these nodes. If we track which support points are feasible, we can use this information to prune the search due to a lack of support for *any* model defined by the points in those nodes.

The key idea behind the variable tree search is that we can use a *dynamic* representation of the potential support. Specifically, we can place the support points in trees and maintain a dynamic *list* of currently valid support nodes. As shown in Figure 1.B, by only testing entire nodes (instead of individual points), we are using spatial coherence of the support points to remove the expense of testing each support point at each step in the search. And by maintaining a list of support tree nodes, we are no longer branching the search over these trees. Thus we remove the need to make a hard "left or right" decision. Further, using a combination of a list and a tree for our representation allows us to refine our support representation on the fly. If we reach a point in the search where a support node is no longer valid, we can simply drop it off the list. And if we reach a point where a support node provides too coarse a representation of the current support space, we can simply remove it and add both of its children to the list.

This leaves the question of when to split support nodes. If we split them too soon, we may end up with many support nodes in our list and mitigate the benefits of the nodes' spatial coherence. If we wait too long to split them, then we may have a few large support nodes that cannot efficiently be pruned. Although we are still investigating splitting strategies, the experiments in this paper use a heuristic that seeks to provide a small number of support nodes that are a reasonable fit to the feasible region. We effectively split a support node if doing so would allow one of its two children to be pruned. For KD-trees this roughly means checking whether the split value lies outside the feasible region.

The full variable tree algorithm is given in Figure 2. A simple example of finding *linear* tracks while using the track's endpoints (earliest and latest in time) as model points and

| | **Variable Tree Model Detection** |
|---|---|
| | **Input**: A set of $M$ current model tree nodes $\mathbf{M}$ |
| | A set of current support tree nodes $\mathbf{S}$ |
| | **Output**: A list $\mathbf{Z}$ of feasible sets of points |
| 1. | $\mathbf{S}' \leftarrow \{\}$ and $\mathbf{S}_{curr} \leftarrow \mathbf{S}$ |
| 2. | IF we cannot prune based on the mutual compatibility of $\mathbf{M}$: |
| 3. | FOR each $\mathbf{s} \in \mathbf{S}_{curr}$ |
| 4. | IF $\mathbf{s}$ is compatible with $\mathbf{M}$: |
| 5. | IF $\mathbf{s}$ is "too wide": |
| 6. | Add $\mathbf{s}$'s left and right child to the end of $\mathbf{S}_{curr}$. |
| 7. | ELSE |
| 8. | Add $\mathbf{s}$ to $\mathbf{S}'$. |
| 9. | IF we have enough valid support points: |
| 10. | IF all of $\mathbf{m} \in \mathbf{M}$ are leaves: |
| 11. | Test all combinations of points owned by the model nodes, using the support nodes' points as potential support. Add valid sets to $\mathbf{Z}$. |
| 12. | ELSE |
| 13. | Let $\mathbf{m}^*$ be the non-leaf model tree node that owns the most points. |
| 14. | Search using $\mathbf{m}^*$'s left child in place of $\mathbf{m}^*$ and $\mathbf{S}'$ instead of $\mathbf{S}$. |
| 15. | Search using $\mathbf{m}^*$'s right child in place of $\mathbf{m}^*$ and $\mathbf{S}'$ instead of $\mathbf{S}$. |

Figure 2: A simple variable tree algorithm for spatial structure search. This algorithm shown uses simple heuristics such as: searching the model node with the most points and splitting a support node if it is too wide. These heuristics can be replaced by more accurate, problem-specific ones.

using all other points for support is illustrated in Figure 3. The first column shows all the tree nodes that are currently part of the search. The second and third columns show the search's position on the two model trees and the current set of valid support nodes respectively. Unlike the pure multiple tree search, the variable tree search does not "branch off" on the support trees, allowing us to consider multiple support nodes from the same time step at any point in the search. Again, it is important to note that by testing the support points as we search, we are both incorporating support information into the pruning decisions and "pruning" the support points for entire sets of models at once.

## 5  Results on the Asteroid Linking Domain

The goal of the single-night asteroid linkage problem is to find sets of 2-dimensional point detections that correspond to a roughly linear motion model. In the below experiments we are interested in finding sets of at least 7 detections from a sequence of 8 images. The movements were constrained to have a speed between 0.05 and 0.5 degrees per day and were allowed an observational error threshold of 0.0003 degrees. All experiments were run on a dual 2.5 GHz Apple G5 with 4 GB of RAM.

The asteroid detection data consists of detections from 8 images of the night sky separated by half-hour intervals. The images were obtained with the MegaCam instrument on the 3.6-meter Canada-France-Hawaii Telescope. The detections, along with confidence levels, were automatically extracted from the images. We can pre-filter the data to pull out only those observations above a given confidence threshold $\sigma$. This allows us to examine how the algorithms perform as we begin to look for increasingly faint asteroids. It should be noted that only limited preprocessing was done to the data, resulting in a very high level

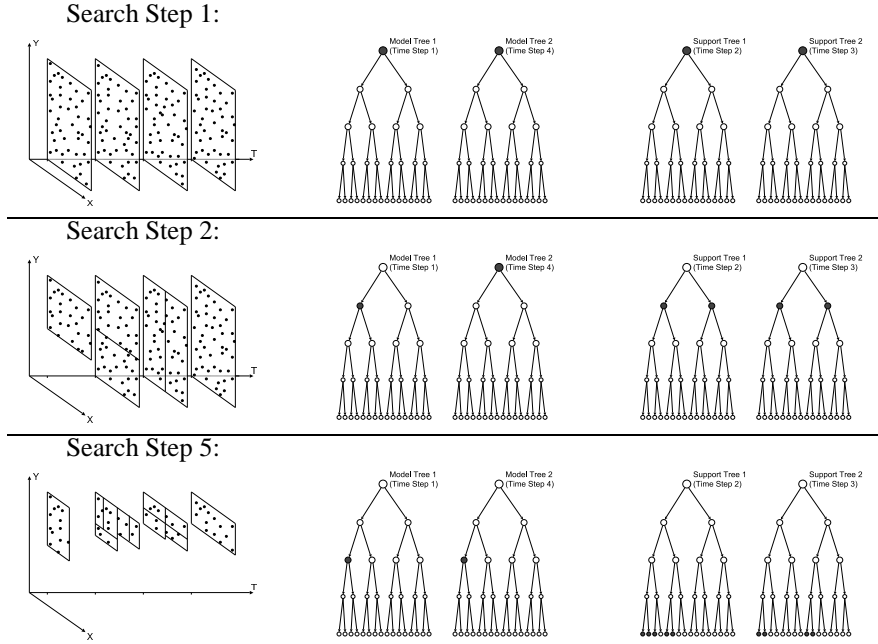

Figure 3: The variable tree algorithm performs a depth first search over the model nodes. At each level of the search the model nodes are checked for mutual compatibility and each support node on the list is check for compatibility with the *set* of model nodes. Since we are not branching on the support nodes, we can split a support node and add *both* children to our list. This figure shows the current model and support nodes and their spatial regions.

Table 1: The running times (in seconds) for the asteroid linkers with different detection thresholds σ and thus different numbers *N* and density of observations.

| σ | 10.0 | 8.0 | 6.0 | 5.0 | 4.0 |
|---|---|---|---|---|---|
| *N* | 3531 | 5818 | 12911 | 24068 | 48646 |
| Single Tree | 2 | 7 | 61 | 488 | 2442 |
| Multiple Tree | 1 | 3 | 30 | 607 | 4306 |
| Variable Tree | < 1 | 1 | 4 | 40 | 205 |

of false detections. While future data sets will contain significantly reduced noise, it is interesting to examine the performance of the algorithms on this real-world high noise, high density data.

The results on the intra-night asteroid tracking domain, shown in Table 1, illustrate a clear advantage to using a variable tree approach. As the significance threshold σ decreases, the number and density of detections increases, allowing the support tree nodes to capture feasibility information for a large number of support points. In contrast, neither the full multiple tree algorithm nor the single-tree algorithm performed well. For the multiple tree algorithm, this decrease in performance is likely due to a combination of the high number of time steps, the allowance of a missing observation, and the high density. In particular, the increased density can reduce opportunities for pruning, causing the algorithm to explore deeper before backtracking.

Table 2: Average running times (in seconds) for a 2-dimensional rectangle search with different numbers of points $N$. The brute force algorithm was only run to $N = 2500$.

| N | 500 | 1000 | 2000 | 2500 | 5000 | 10000 | 25000 | 50000 |
|---|---|---|---|---|---|---|---|---|
| Brute Force | 0.37 | 2.73 | 21.12 | 41.03 | n/a | n/a | n/a | n/a |
| Single Tree | 0.02 | 0.07 | 0.30 | 0.51 | 2.15 | 10.05 | 66.24 | 293.10 |
| Multi-Tree | 0.01 | 0.02 | 0.06 | 0.09 | 0.30 | 1.11 | 6.61 | 27.79 |
| Variable-Tree | 0.01 | 0.02 | 0.05 | 0.07 | 0.22 | 0.80 | 4.27 | 16.30 |

Table 3: Average running times (in seconds) for a rectangle search with different numbers of required corners $k$. For this experiment $N = 10000$ and $D = 3$.

| k | 8 | 7 | 6 | 5 | 4 |
|---|---|---|---|---|---|
| Single Tree | 4.71 | 4.72 | 4.71 | 4.71 | 4.71 |
| Multi-Tree | 3.96 | 19.45 | 45.02 | 67.50 | 78.81 |
| Variable-Tree | 0.65 | 0.75 | 0.85 | 0.92 | 1.02 |

## 6  Experiments on the Simulated Rectangle Domain

We can apply the above techniques to a range of other model-based spatial search problems. In this section we consider a toy template matching problem, finding axis-aligned hyper-rectangles in $D$-dimensional space by finding $k$ or more *corners* that fit a rectangle. We use this simple, albeit artificial, problem both to demonstrate potential pattern recognition applications and to analyze the algorithms as we vary the properties of the data.

Formally, we restrict the model to use the upper and lower corners as the two model points. Potential support points are those points that fall within some threshold of the other $2^D - 2$ corners. In addition, we restrict the allowable bounds of the rectangles by providing a maximum width.

To evaluate the algorithms' relative performance, we used random data generated from a uniform distribution on a unit hyper-cube. The threshold and maximum width were fixed for all experiments at 0.0001 and 0.2 respectively. All experiments were run on a dual 2.5 GHz Apple G5 with 4 GB of RAM.

The first factor that we examined was how each algorithm scales with the number of points. We generated random data with 5 known rectangles and $N$ additional random points and computed the average wall-clock running time (over ten trials) for each algorithm. The results, shown in Table 2, show a graceful scaling of all of the multiple tree algorithms. In contrast, the brute force and single tree algorithms run into trouble as the number of points becomes moderately large. The variable tree algorithm consistently performs the best, as it is able to avoid significant amounts of redundant computation.

One potential drawback of the full multiple tree algorithm is that since it branches on all points, it may become inefficient as the allowable number of missing support points grows. To test this we looked at 3-dimensional data and varied the minimum number of required support points $k$. As shown in Table 3, all multiple tree methods become *more* expensive as the number of required support points decreases. This is especially the case for the multi-tree algorithm, which has to perform several almost identical searches to account for missing points. However, the variable-tree algorithm's performance degrades gracefully and is the best for all trials.

# 7 Conclusions

Tree-based spatial algorithms provide the potential for significant computational savings with multiple tree algorithms providing further opportunities to exploit structure in the data. However, a distinct trade-off exists between ignoring structure from all aspects of the problem and increasing the combinatorics of the search. We presented a variable tree approach that exploits the advantages of both single tree and multiple tree algorithms. A combinatorial search is carried out over just the minimum number of model points, while still tracking the feasibility of the various support points. As shown in the above experiments, this approach provides significant computational savings over both the traditional single tree and and multiple tree searches. Finally, it is interesting to note that the dynamic support technique described in this paper is general and may be applied to a range of other algorithms, such as the Fast Hough Transform [10], that maintain information on which points support a given model.

### Acknowledgments

Jeremy Kubica is supported by a grant from the Fannie and John Hertz Foundation. Andrew Moore and Andrew Connolly are supported by a National Science Foundation ITR grant (CCF-0121671).

## Footnotes

[1]Typically $M = c$, although in some cases it may be beneficial to use a different number of model nodes.

# References

[1] A.J. Storkey, N.C. Hambly, C.K.I. Williams, and R.G. Mann. Renewal Strings for Cleaning Astronomical Databases. In *UAI 19*, 559-566, 2003.

[2] M.A. Fischler and R.C. Bolles. Random Sample Consensus: A Paradigm for Model Fitting with Applications to Image Analysis and Automated Cartography. *Comm. of the ACM*, 24:381–395, 1981.

[3] S. Blackman and R. Popoli. *Design and Analysis of Modern Tracking Systems*. Artech House, 1999.

[4] J.L. Bentley . Multidimensional Binary Search Trees Used for Associative Searching. *Comm. of the ACM*, 18 (9), 1975.

[5] J. K. Uhlmann. Algorithms for multiple-target tracking. *American Scientist*, 80(2):128–141, 1992.

[6] P. V. C. Hough. Machine analysis of bubble chamber pictures. In *International Conference on High Energy Accelerators and Instrumentation*. CERN, 1959.

[7] A. Gray and A. Moore. N-body problems in statistical learning. In T. K. Leen and T. G. Dietterich, editors, *Advances in Neural Information Processing Systems*. MIT Press, 2001.

[8] G. R. Hjaltason and H. Samet. Incremental distance join algorithms for spatial databases. In *Proc. of the 1998 ACM-SIGMOD Conference*, 237–248, 1998.

[9] J. Kubica, A. Moore, A. Connolly, and R. Jedicke. A Multiple Tree Algorithm for the Efficient Association of Asteroid Observations. In *KDD'05*. August 2005.

[10] H. Li, M.A. Lavin, and R.J. Le Master. Fast Hough Transform: A Hierarchical Approach. In *Computer Vision, Graphics, and Image Processing*, 36(2-3):139–161, November 1986.
